# Learning Temporal Dependencies in Connectionist Speech Recognition

**Steve Renals**  **Mike Hochberg**  **Tony Robinson**
Cambridge University Engineering Department
Cambridge CB2 1PZ, UK
{sjr,mmh,ajr}@eng.cam.ac.uk

## Abstract

Hybrid connectionist/HMM systems model time both using a Markov chain and through properties of a connectionist network. In this paper, we discuss the nature of the time dependence currently employed in our systems using recurrent networks (RNs) and feed-forward multi-layer perceptrons (MLPs). In particular, we introduce local recurrences into a MLP to produce an enhanced input representation. This is in the form of an adaptive gamma filter and incorporates an automatic approach for learning temporal dependencies. We have experimented on a speaker-independent phone recognition task using the TIMIT database. Results using the gamma filtered input representation have shown improvement over the baseline MLP system. Improvements have also been obtained through merging the baseline and gamma filter models.

## 1  INTRODUCTION

The most common approach to large-vocabulary, talker-independent speech recognition has been statistical modelling with hidden Markov models (HMMs). The HMM has an explicit model for time specified by the Markov chain parameters. This temporal model is governed by the grammar and phonology of the language being modelled. The acoustic signal is modelled as a random process of the Markov chain and adjoining local temporal information is assumed to be independent. This assumption is certainly not the case and a great deal of research has addressed the problem of modelling acoustic context.

Standard HMM techniques for handling the context dependencies of the signal have ex-

plicitly modelled all the n-tuples of acoustic segments (*e.g.*, context-dependent triphone models). Typically, these systems employ a great number of parameters and, subsequently, require massive amounts of training data and/or care in smoothing of the parameters. Where the context of the model is greater than two segments, an additional problem is that it is very likely that contexts found in testing data are never observed in the training data.

Recently, we have developed state-of-the-art continuous speech recognition systems using hybrid connectionist/HMM methods (Robinson, 1994; Renals et al., 1994). These hybrid connectionist/HMM systems model context at two levels (although these levels are not necessarily at distinct scales). As in the traditional HMM, a Markov process is used to specify the duration and lexical constraints on the model. The connectionist framework provides a conditional likelihood estimate of the local (in time) acoustic waveform given the Markov process. Acoustic context is handled by either expanding the network input to include multiple, adjacent input frames, or using recurrent connections in the network to provide some memory of the previous acoustic inputs.

## 2   DEPTH AND RESOLUTION

Following Principe et al. (1993), we may characterise the time dependence displayed by a particular model in terms of *depth* and *resolution*. Loosely speaking, the depth tells us how far back in time a model is able to look[1], and the resolution tells us how accurately the past to a given depth may be reconstructed. The baseline models that we currently use are very different in terms of these characteristics.

### Multi-layer Perceptron

The feed-forward multi-layer perceptron (MLP) does not naturally model time, but simply maps an input to an output. Crude temporal dependence may be imparted into the system by using a delay-lined input (figure 1a); an extension of this approach is the time-delay neural network (TDNN). The MLP may be interpreted as acting as a FIR filter. A delay-lined input representation may be characterised as having low depth (limited by the delay line length) and high resolution (no smoothing).

### Recurrent Network

The recurrent network (RN) models time dependencies of the acoustic signal via a fully-connected, recurrent hidden layer (figure 1b). The RN has a potentially infinite depth (although in practice this is limited by available training algorithms) and low resolution, and may be regarded as analogous to an IIR filter. A small amount of future context is available to the RN, through a four frame target delay.

### Experiments

Experiments on the DARPA Resource Management (RM) database have indicated that the tradeoff between depth and resolution is important. In Robinson et al. (1993), we compared different acoustic front ends using a MLP and a RN. Both networks used 68

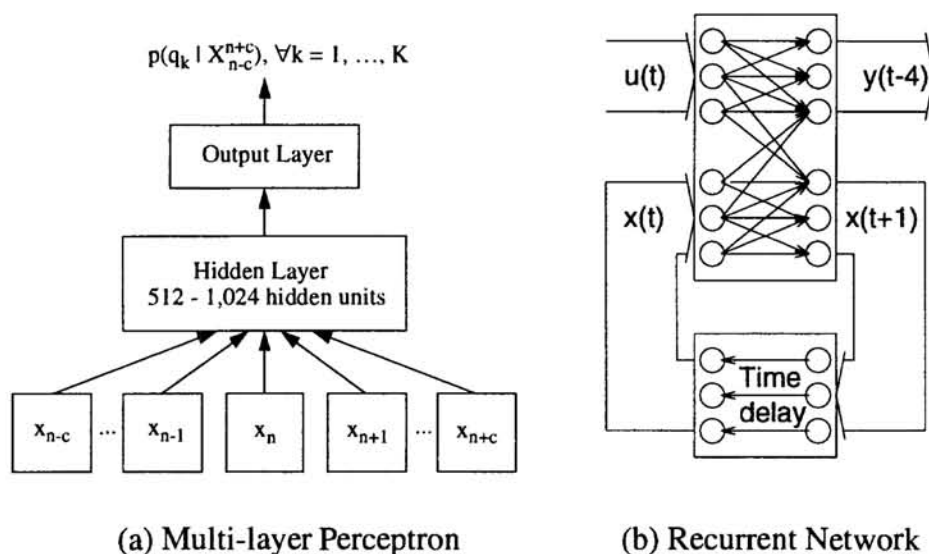

(a) Multi-layer Perceptron                    (b) Recurrent Network

Figure 1: Connectionist architectures used for speech recognition.

outputs (corresponding to phones); the MLP used 1000 hidden units and the RN used 256 hidden units. Both architectures were trained using a training set containing 3990 sentences spoken by 109 speakers. Two different resolutions were used in the front-end computation of mel-frequency cepstral coefficients (MFCCs): one with a 20ms Hamming window and a 10ms frame step (referred to as 20/10), the other with a 32ms Hamming window and a 16ms frame step (referred to as 32/16). *A priori*, we expected the higher resolution frame rate (20/10) to produce a higher performance recogniser because rapid speech events would be more accurately modelled. While this was the case for the MLP, the RN showed better results using the lower resolution front end (32/16) (see table 1). For the higher resolution front-end, both models require a greater depth (in frames) for the same context (in milliseconds). In these experiments the network architectures were constant so increasing the resolution of the front end results in a loss of depth.

| Net | Front End | Word Error Rate % | | | |
|---|---|---|---|---|---|
| | | feb89 | oct89 | feb91 | sep92 |
| RN | 20/10 | 6.1 | 7.6 | 7.4 | 12.1 |
| RN | 32/16 | 5.9 | 6.3 | 6.1 | 11.5 |
| MLP | 20/10 | 5.7 | 7.1 | 7.6 | 12.0 |
| MLP | 32/16 | 6.6 | 7.8 | 8.5 | 15.0 |

Table 1: Comparison of acoustic front ends using a RN and a MLP for continuous speech recognition on the RM task, using a wordpair grammar of perplexity 60. The four test sets (feb89, oct89, feb91 and sep92, labelled according to their date of release by DARPA) each contain 300 sentences spoken by 10 new speakers.

In the case of the MLP we were able to explicitly set the memory depth. Previous experiments had determined that a memory depth of 6 frames (together with a target delayed by 3 frames) was adequate for problems relating to this database. In the case of the RN, memory

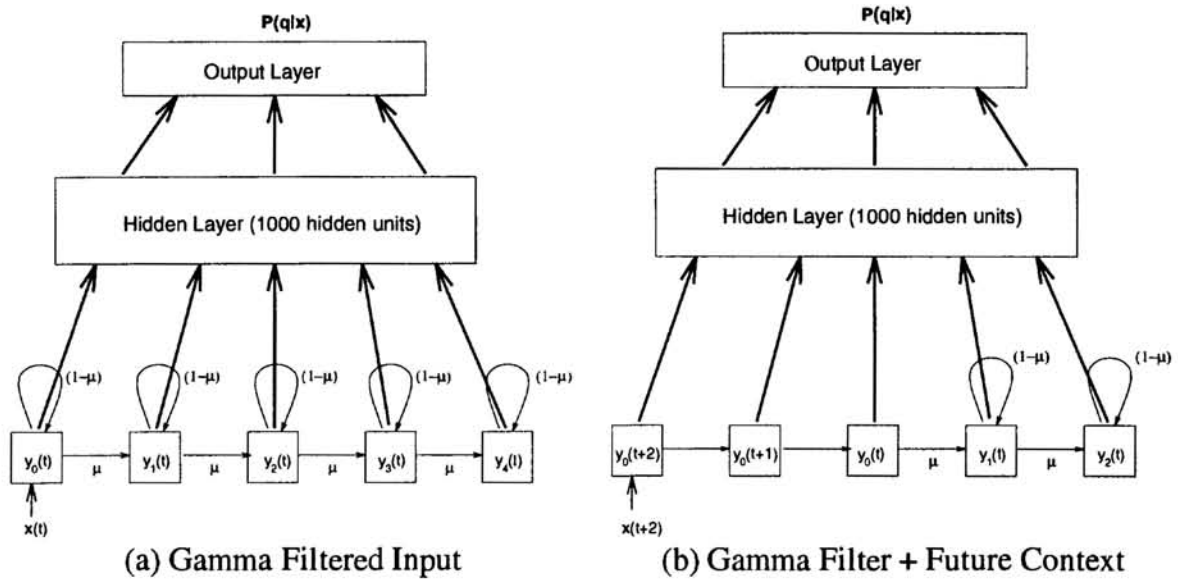

(a) Gamma Filtered Input          (b) Gamma Filter + Future Context

Figure 2: Gamma memory applied to the network input. The simple gamma memory in (a) does not incorporate any information about the future, unless the target is delayed. In (b) there is an explicit delay line to incorporate some future context.

depth is not determined directly, but results from the interaction between the network architecture (*i.e.*, number of state units) and the training process (in this case, back-propagation through time). We hypothesise that the RN failed to make use of the higher resolution front end because it did not adapt to the required depth.

## 3   GAMMA MEMORY STRUCTURE

The tradeoff between depth and resolution has led us to investigate other network architectures. The gamma filter, introduced by de Vries and Principe (1992) and Principe et al. (1993), is a memory structure designed to automatically determine the appropriate depth and resolution (figure 2). This locally recurrent architecture enables lowpass and bandpass filters to be learned from data (using back-propagation through time or real-time recurrent learning) with only a few additional parameters.

We may regard the gamma memory as a generalisation of a delay line (Mozer, 1993) in which the $k$th tap at time $t$ is obtained by convolving the input time series with a kernel function, $g_k^\mu(t)$, and where $\mu$ parametrises the $K$th order gamma filter,

$$g_0^\mu(t) = \delta(t)$$

$$g_k^\mu(t) = \frac{\mu^k}{(k-1)!}t^{k-1}e^{-\mu t} \quad 1 \le k < K .$$

This family of kernels is attractive, since it may be computed incrementally by

$$\frac{dx_k(t)}{dt} = -\mu x_k(t) + \mu x_{k-1}(t) .$$

This is in contrast to some other kernels that have been proposed (*e.g.*, Gaussian kernels proposed by Bodenhausen and Waibel (1991) in which the convolutions must be performed

explicitly). In the discrete time case the filter becomes:

$$x_k(t) = (1 - \mu)x_k(t - 1) + \mu x_{k-1}(t - 1) \ .$$

This recursive filter is guaranteed to be stable when $0 < \mu < 2$.

In the experiments reported below we have replaced the input delay line of a MLP with a gamma memory structure, using one gamma filter for each input feature. This structure is referred to as a "focused gamma net" by de Vries and Principe (1992).

Owing to the effects of anticipatory coarticulation, information about the future is as important as past context in speech recognition. A simple gamma filtered input (figure 2a) does not include any future context. There are various ways in which this may be remedied;

- Use the same architecture, but delay the target (similar to figure 1b);
- Explicitly specify future context by adding a delay line from the future (figure 2b);
- Use two gamma filters per feature: one forward, one backward in time.

A drawback of the first approach is that the central frame corresponding to the delayed target will have been smoothed by the action of the gamma filter. The third approach necessitates two passes when either training or running the network.

## 4   SPEECH RECOGNITION EXPERIMENTS

We have performed experiments using the standard TIMIT speech database. This database is divided into 462 training speakers and 168 test speakers. Each speaker utters eight sentences that are used in these experiments, giving a training set of 3696 sentences and a test set of 1344 sentences. We have used this database for a continuous phone recognition task: labelling each sentence using a sequence of symbols, drawn from the standard 61 element phone set.

The acoustic data was preprocessed using a 12th order perceptual linear prediction (PLP) analysis to produce an energy coefficient plus 12 PLP cepstral coefficients for each frame of data. A 20ms Hamming window was used with a 10ms frame step. The temporal derivatives of each of these features was also estimated (using a linear regression over $\pm$ 3 adjacent frames) giving a total of 26 features per frame.

The networks we employed (table 2) were MLPs, with 1000 hidden units, 61 output units (one per phone) and a variety of input representations. The Markov process used single state phone models, a bigram phone grammar, and a Viterbi decoder was used for recognition. The feed-forward weights in each network were initialised with identical sets of small random values. The gamma filter coefficients were initialised to 1.0 (equivalent to a delay line). The feed-forward weights were trained using back-propagation and the gamma filter coefficients were trained in a forward in time back-propagation procedure equivalent to real-time recurrent learning. An important detail is that the gradient step size was substantially lower (by a factor of 10) for the gamma filter parameters compared with the feed-forward weights. This was necessary to prevent the gamma filter parameters from becoming unstable.

The baseline system using a delay line (**Base**) corresponds to figure 1a, with $\pm$ 3 frames of context. The basic four-tap gamma filter **G4** is illustrated in figure 2a (but using 1 fewer

| System ID | Description |
|---|---|
| Base | Baseline delay line, ± 3 frames of context |
| G4 | Gamma filter, 4 taps |
| G7 | Gamma filter, 7 taps, delayed target |
| G7i | G7 initialised using weights from Base |
| G4F3 | Gamma filter, 4 taps, 3 frames future context |
| G4F3i | G4F3 initialised using weights from Base |

Table 2: Input representations used in the experiments. Note that G7i and G4F3i were initialised using a partially trained weight matrix (after six epochs) from Base.

tap than the picture) and G7 is a 7 frame gamma filter with the target delayed for 3 frames, thus providing some future context (but at the expense of smoothing the "centre" frame). Future context is explicitly incorporated in G4F3, in which the three adjacent future frames are included (similar to figure 2b). Systems G7i and G4F3i were both initialised using a partially trained weight matrix for the delay line system, Base. This was equivalent to fixing the value of the gamma filter coefficients to a constant (1.0) during the first six epochs of training and only adapting the feed-forward weights, before allowing the gamma filter coefficients to adapt.

The results of using these systems on the TIMIT phone recognition task are given in table 3. Table 4 contains the results of some model merging experiments, in which the output probability estimates of 2 or more networks were averaged to produce a merged estimate.

| System ID | Depth | Correct% | Insert.% | Subst.% | Delet.% | Error% |
|---|---|---|---|---|---|---|
| Base | 4.0 | 67.6 | 4.1 | 24.7 | 7.7 | 36.5 |
| G4 | 8.5 | 65.8 | 4.1 | 25.9 | 8.3 | 38.2 |
| G7 | 11.7 | 65.5 | 4.1 | 26.0 | 8.5 | 38.6 |
| G7i | 5.8 | 67.3 | 3.8 | 24.5 | 8.2 | 36.5 |
| G4F3 | 9.6 | 67.8 | 3.8 | 24.2 | 8.0 | 36.0 |
| G4F3i | 4.9 | 68.0 | 3.9 | 24.2 | 7.8 | 35.9 |

Table 3: TIMIT phone recognition results for the systems defined in table 2. The Depth value is estimated as the ratio of filter order to average filter parameter $K/\mu$. Future context is ignored in the estimate of depth, and the estimates for G7 and G7i are adjusted to account for the delayed target.

| System ID | Correct% | Insert.% | Subst.% | Delet.% | Error% |
|---|---|---|---|---|---|
| G4F3 + Base | 68.1 | 3.2 | 23.7 | 8.2 | 35.1 |
| G4F3 + G4F3i | 68.2 | 3.2 | 23.5 | 8.3 | 35.0 |
| G7 + Base | 67.0 | 3.2 | 24.4 | 8.6 | 36.2 |
| G7 + G7i | 67.4 | 3.6 | 24.4 | 8.2 | 36.2 |

Table 4: Model merging on the TIMIT phone recognition task.

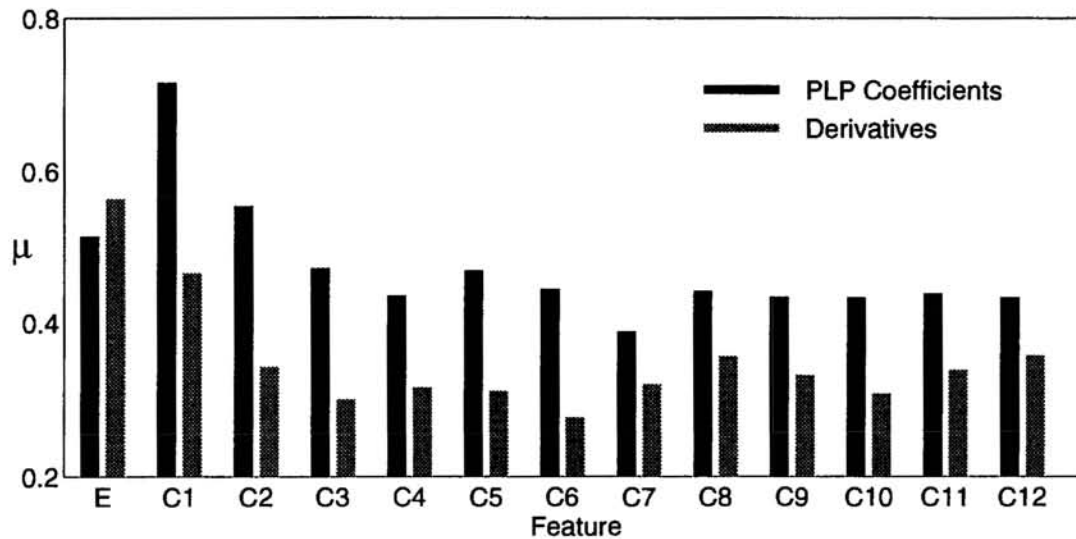

Figure 3: Gamma filter coefficients for **G4F3**. The coefficients correspond to energy (*E*) and 12 PLP cepstral coefficients (*C*1–*C*12) and their temporal derivatives.

## 5   DISCUSSION

Several comments may be made about the results in section 4. As can be seen in table 3, replacing a delay line with an adaptive gamma filter can lead to an improvement in performance. Knowledge of future context is important. This is shown by **G4**, which had no future context or delayed target information, and had poorer performance than the baseline. However, incorporating future context using a delay line (**G4F3**) gives better performance than a pure gamma filter representation with a delayed target (**G7**). Training the locally recurrent gamma filter coefficients is not trivial. Fixing the gamma filter coefficients to 1.0 (delay line) whilst adapting the feed-forward weights during the first part of training is beneficial. This is demonstrated by comparing the performance of **G7** with **G7i** and **G4F3** with **G4F3i**. Finally, table 4 shows that model merging generally leads to improved recognition performance relative to the component models. This also indicates that the delay line and gamma filter input representations are somewhat complementary.

Figure 3 displays the trained gamma filter coefficients for **G4F3**. There are several points to make about the learned temporal dependencies.

- The derivative parameters are smaller compared with the static PLP parameters. This indicates the derivative filters have greater depth and lower resolution compared with the static PLP filters.

- If a gamma filter is regarded as a lowpass IIR filter, then lower filter coefficients indicate a greater degree of smoothing. Better estimated coefficients (*e.g.*, static PLP coefficients C1 and C2) give rise to gamma filters with less smoothing.

- The training schedule has a significant effect on filter coefficients. The depth estimates of **G4F3** and **G4F3i** in table 3 demonstrate that very different sets of filters were arrived at for the same architecture with identical initial parameters, but with different training schedules.

We are investigating the possibility of using gamma filters to model speaker characteristics. Preliminary experiments in which the gamma filters of speaker independent networks were adapted to a new speaker have indicated that the gamma filter coefficients are speaker dependent. This is an attractive approach to speaker adaptation, since very few parameters (26 in our case) need be adapted to a new speaker.

Gamma filtering is a simple, well-motivated approach to modelling temporal dependencies for speech recognition and other problems. It adds minimal complexity to the system (in our case a parameter increase of 0.01%), and these initial experiments have shown an improvement in phone recognition performance on the TIMIT database. A further increase in performance resulted from a model merging process. We note that gamma filtering and model merging may be regarded as two sides of the same coin: gamma filtering smooths the input acoustic features, while model merging smooths the output probability estimates.

## Acknowledgement

This work was supported by ESPRIT BRA 6487, WERNICKE. SR was supported by a SERC postdoctoral fellowship and a travel grant from the NIPS foundation. TR was supported by a SERC advanced fellowship.

## Footnotes

[1]In the language of section 3, the depth may be expressed as the mean duration, relative to the target, of the last kernel in a filter that is convolved with the input.

## References

Bodenhausen, U., & Waibel, A. (1991). The Tempo 2 algorithm: Adjusting time delays by supervised learning. In Lippmann, R. P., Moody, J. E., & Touretzky, D. S. (Eds.), *Advances in Neural Information Processing Systems*, Vol. 3, pp. 155–161. Morgan Kaufmann, San Mateo CA.

de Vries, B., & Principe, J. C. (1992). The gamma model—a new neural model for temporal processing. *Neural Networks, 5*, 565–576.

Mozer, M. C. (1993). Neural net architectures for temporal sequence processing. In Weigend, A. S., & Gershenfeld, N. (Eds.), *Predicting the future and understanding the past*. Addison-Wesley, Redwood City CA.

Principe, J. C., de Vries, B., & de Oliveira, P. G. (1993). The gamma filter—a new class of adaptive IIR filters with restricted feedback. *IEEE Transactions on Signal Processing, 41*, 649–656.

Renals, S., Morgan, N., Bourlard, H., Cohen, M., & Franco, H. (1994). Connectionist probability estimators in HMM speech recognition. *IEEE Transactions on Speech and Audio Processing*. In press.

Robinson, A. J., Almeida, L., Boite, J.-M., Bourlard, H., Fallside, F., Hochberg, M., Kershaw, D., Kohn, P., Konig, Y., Morgan, N., Neto, J. P., Renals, S., Saerens, M., & Wooters, C. (1993). A neural network based, speaker independent, large vocabulary, continuous speech recognition system: the WERNICKE project. In *Proceedings European Conference on Speech Communication and Technology*, pp. 1941–1944 Berlin.

Robinson, T. (1994). The application of recurrent nets to phone probability estimation. *IEEE Transactions on Neural Networks*. In press.
